# Inducing Metric Violations in Human Similarity Judgements

**Julian Laub[1], Jakob Macke[2], Klaus-Robert Müller[1,3] and Felix A. Wichmann[2]**
[1] Fraunhofer FIRST.IDA, Kekulestr. 7, 12489 Berlin, Germany
[2] Max Planck Institut for Biological Cybernetics, Spemannstr. 38, 72076 Tübingen, Germany
[3] University of Potsdam, Department of Computer Science
August-Bebel-Strasse 89, 14482 Potsdam, Germany
{jlaub,klaus}@first.fhg.de
{felix,jakob}@tuebingen.mpg.de

## Abstract

Attempting to model human categorization and similarity judgements is both a very interesting but also an exceedingly difficult challenge. Some of the difficulty arises because of conflicting evidence whether human categorization and similarity judgements should or should not be modelled as to operate on a mental representation that is essentially metric. Intuitively, this has a strong appeal as it would allow (dis)similarity to be represented geometrically as distance in some internal space. Here we show how a *single* stimulus, carefully constructed in a psychophysical experiment, introduces $l_2$ violations in what used to be an internal similarity space that could be adequately modelled as Euclidean. We term this one influential data point a *conflictual judgement*. We present an algorithm of how to analyse such data and how to identify the crucial point. Thus there may not be a strict dichotomy between either a metric or a non-metric internal space but rather degrees to which potentially large subsets of stimuli are represented metrically with a small subset causing a global violation of metricity.

## 1  Introduction

The central aspect of quantitative approaches in psychology is to adequately model human behaviour. In perceptual research, for example, all successful models of visual perception tacitly assume that at least simple visual stimuli are processed, transformed and compared to some internal reference in a metric space. In cognitive psychology many models of human categorisation, too, assume that stimuli "similar" to each other are grouped together in categories. Within a category similarity is very high whereas between categories similarity is low. This coincides with intuitive notions of categorization which, too, tend to rely on similarity despite serious problems in defining what similarity means or ought to mean [6]. Work on similarity and generalization in psychology has been hugely influenced by the work of Roger Shepard on similarity and categorization [12, 14, 11, 4, 13]. Shepard explicitly assumes that similarity is a distance measure in a metric space, and many perceptual categorization models follow Shepard's general framework [8, 3].

This notion of similarity is frequently linked to a geometric representation where stimuli are points in a space and the similarity is linked to an intuitive metric on this space, e.g. the Euclidean metric. In a well-known and influential series of papers Tversky and colleagues have challenged the idea of a geometric representation of similarity, however [16, 17]. They provided convincing evidence that (intuitive, and certainly Euclidean) geometric representations cannot account for human similarity judgements—at least for the highly cognitive and *non-perceptual* stimuli they employed in their studies. Within their experimental context pairwise dissimilarity measurements violated metricity, in particular symmetry and the triangle inequality. Technically, violations of Euclideanity translate

into non positive semi-definite similarity matrices ("pseudo-Gram" matrices) [15], a fact, which imposes severe constraints on the data analysis procedures. Typical approaches to overcome these problems involve leaving out negative eigenvalues altogether or shifting the spectrum for subsequent (Kernel-)PCA analysis [10, 7]. The shortcomings of such methods are that they assume that the data really are Euclidean and that all violations are only due to noise.

Shepard's solution to non-metricity was to find non-linear transformations of the similarity data of the subjects to make them Euclidean, and/or use non-Euclidean metrics such as the city-block metric (or other Minkowski p-norms with $p \neq 2$)[11, 4]. Yet another way how metric violations may arise in experimental data—whilst retaining the notion that the internal, mental representation is really metric—is to invoke attentional re-weighting of dimensions during similarity judgements and categorisation tasks [1]. Here we develop a position in between the seeming dichotomy of "metric versus non-metric" internal representations: Our alternative and complementary suggestion is that a potentially very small subset—in fact a single observation or data point or stimulus—of the data may induce the non-metricity, or at least a non-Euclidean metric: in a theoretical setting it has been shown that systematic violation of metricity can be due to an *interesting* subset of the data—i.e. not due to noise [5]. We show how *conflictual judgments* can introduce metric violation in a situation where the human similarity judgments are based upon smooth geometric features and are otherwise essentially Euclidean.

First we present a simple model which explains the occurrence of metric violations in similarity data, with a special focus on human similarity judgments. Thereafter both models are tested with data obtained from psychophysical experiments specifically designed to induce *conflictual judgments*.

## 2   Modeling metric violations for single conflictual situations

A dissimilarity function $d$ is called *metric* if: $d(x_i, x_j) \geqslant 0 \ \forall \ x_i, x_j \in X, d(x_i, x_j) = 0$ iff $x_i = x_j$, $d(x_i, x_j) = d(x_j, x_i) \ \forall \ x_i, x_j \in X, d(x_i, x_k) + d(x_k, x_j) \geqslant d(x_i, x_j) \ \forall \ x_i, x_j, x_k \in X$. A dissimilarity matrix $D = (D_{ij})$ will be called *metric* if there exists a metric $d$ such that $D_{ij} = d(\cdot, \cdot)$. $D = (D_{ij})$ will be called squared Euclidean if the metric derives from $l_2$.

It can be shown that $D$ is $l_2$ (Euclidean) iff $C = -\frac{1}{2}QDQ$ is positive semi-definite ($Q = I - \frac{1}{n}ee'$ be the projection matrix on the orthogonal complement of $e = (1, 1, \dots 1)'$). $C$ is called the Gram matrix. An indefinite $C$ will be called a *pseudo-Gram matrix*. A non-metric $D$ is, a fortiori, non $l_2$ and thus its associated $C$ is indefinite. On the other hand, when $C$ is indefinite, we can conclude that $D$ is non $l_2$, but not necessarily non-metric. Non-metricity of $D$ must be verified by testing the above four requirements.

We now introduce a simple model for conflictual human similarity. Let $\{f_1, f_2, \dots f_n\}$ be a basis. A given data point $x_i$ can be decomposed in this basis as $x_i = \sum_{k=1}^{n} \alpha_k^{(i)} f_k$. The squared $l_2$ distance between $x_i$ and $x_j$ therefore reads: $d_{ij} = ||x_i - x_j||^2 = \left|\left| \sum_{k=1}^{n} \left(\alpha_k^{(i)} - \alpha_k^{(j)}\right) f_k \right|\right|^2$. However this assumes constant feature-perception, i.e. a constant mental image with respect to different tasks. In the realm of human perception this is not always the case, as illustrated by the following well known ambiguous figure (Fig. 1). We hypothesise that the ambiguous perception of such figures corresponds to some kind of "perceptual state-switching". If the state-switching could be experimentally induced within a single experiment and subject, this may cause metric or at least Euclidean violations by this *conflictual judgment*.

A possible way to model such conflictual situations in human similarity judgments is to introduce states $\{\omega^{(1)}, \omega^{(2)} \dots \omega^{(d)}\}, \omega^{(l)} \in \mathbb{R}^n$ for $l = 1, 2, \dots d$, affecting the features. The similarity judgment between objects then depends on the perceptual state (weight) the subject is in. Assuming that the person is in state $\omega^{(l)}$ the distance becomes: $d_{ij} = ||x_i - x_j||^2 = \left|\left| \sum_{k=1}^{n} \left(\alpha_k^{(i)} - \alpha_k^{(j)}\right) \omega_k^{(l)} f_k \right|\right|^2$. *With no further restriction this model yields non-metric distance matrices.*

$\omega$ may vary between different subjects reflecting their different focus of attention, thus we will not average the similarity judgments over different subjects but only over different trials of one single subject, assuming that for a given person $\omega$ is constant.

In order to interpret the metric violations, we propose the following simple algorithm, which allows to specifically visualize the information coded by the negative eigenvalues. It essentially relies upon

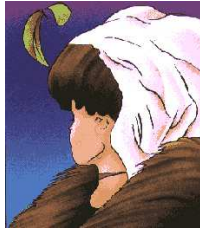
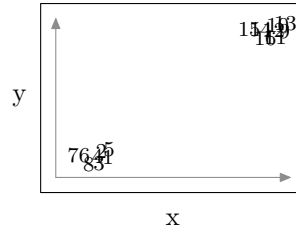

Figure 1: Left: What do you see? A young lady or an old woman? If you were to compare this picture to a large set of images of young ladies or old women, the perceptual state-switch could induce large individual weights on the similarity. Right: Simple data distribution (left) used in the proof of concept illustration in subsection 2.1.

the embedding of non-metric pairwise data into a pseudo-Euclidean space (see [2, 9] and references therein for details):

$$\text{non squared-Euclidean } D \xrightarrow{\ C = -1/2QDQ\ } C \text{ with negative eigenvalues}$$
$$C \xrightarrow{\text{spectral decomposition}} V\Lambda V^\top = V|\Lambda|^{\frac{1}{2}} M |\Lambda|^{\frac{1}{2}} V^\top$$
$$X_P^* = |\Lambda_P|^{1/2} V_P^\top,$$

where $V$ is the column matrix of eigenvectors, $\Lambda$ the diagonal matrix of the corresponding eigenvalues and $M$ the block-matrix consisting of the blocks $I_{p\times p}$, $-I_{q\times q}$ and $0_{k\times k}$ (with $k = n - p - q$) The columns of $X_P^*$ contain the vectors $x_i$ in $p$-dimensional subspace $P$.

Retaining only the first two coordinates ($P = \{v_1, v_2\}$) of the obtained vectors corresponds to a projection onto the first two leading eigendirections. Retaining the last two ($P = \{v_n, v_{n-1}\}$) is a projection onto the last two eigendirections:

*This corresponds to a projection onto directions related to the negative part of $C$ and containing the information coded by the $l_2$ violations.*

## 2.1 Proof of concept illustration: single conflicts introduce metric violations

We now illustrate the model for a single conflictual situation.

Consider a weight $\omega^{(l)}$ constant for all feature-vectors, taken to be the unit vectors $e_k$ in this example. Then we have $d_{ij} = \left(\omega^{l_{ij}}\right)^2 \left\| \sum_{k=1}^n \left(\alpha_k^{(i)} - \alpha_k^{(j)}\right) e_k \right\|^2 = \left(\omega^{l_{ij}}\right)^2 \|x_i - x_j\|_2^2$, where $\|\cdot\|_2$ is the usual unweighted Euclidean norm.

For a simple illustration we take 16 points distributed in two Gaussian blobs (Fig. 1, right) with squared Euclidean distance given by $d_2$ to represent the objects to compare. Suppose an experimental subject is to pairwise compare these objects to give it a dissimilarity score and that a conflictual situation arises for the pairs $(2, 3)$, $(7, 2)$ and $(6, 5)$ translating in a strong weighting of these dissimilarities. For the sake of the example, we chose the (largely exaggerated) weights to be 150, 70 and 220 respectively, acting as follows: $d(2, 3) = d_2(2, 3) \cdot 150$, $d(7, 2) = d_2(7, 2) \cdot 70$, $d(6, 5) = d_2(6, 5) \cdot 220$. The corresponding $d$ is non-Euclidean and its associated $C$ is indefinite. The spectrum of $C$ is given in Fig. 2, right, and exhibits a clear negative spectrum.

Fig. 2 shows the projection onto the leading positive and leading negative eigendirections of the both the unweighted distance (top row) and the weighted distance matrix (bottom row). Both yield the same grouping in the positive part. In the negative eigenspace we obtain a singular distribution for the unweighted case. This is *not* the case for the weighted dissimilarity: we see that the distribution in the negative separates the points whose mutual distance has been (strongly) weighted. The information contained in the negative part, reflecting the information coded by metric or $l_2$ violations, codes in this case for the individual weighting of the (dis)similarities.

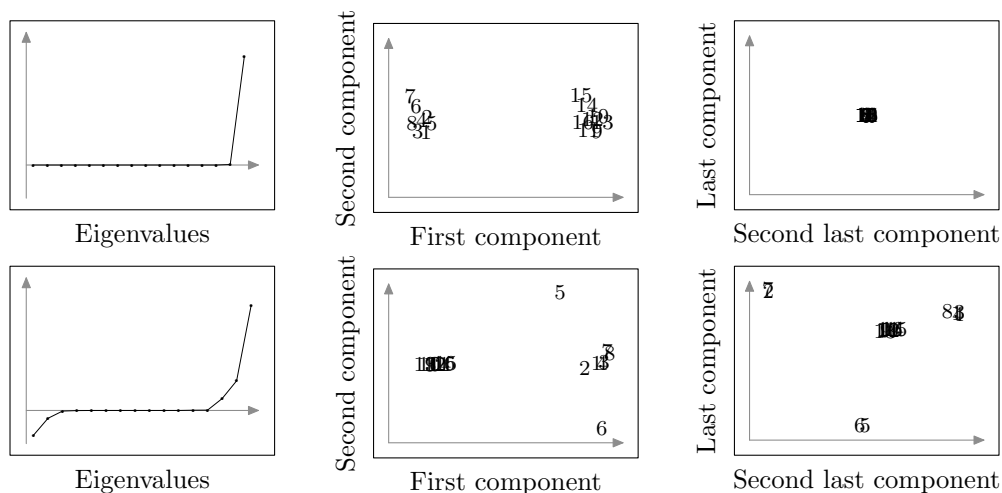

Figure 2: Proof of concept: Unperturbed dissimilarity matrix (no conflict) and weighted dissimilarity matrix (conflict). Single weighting of dissimilarities introduce metric violations and hence $l_2$ violations which reflect in negative spectra. The conflictual points are peripheral in the projection onto the negative eigenspace centered around the bulk of points whose dissimilarities are essentially Euclidean. Note that because of the huge weights, these effects are largely exaggerated in comparison to real world judgments.

## 3   Experiments

Twenty gray-scale 256 x 256-pixel images of faces were generated from the MPI-face database [1]. All faces were normalized to have the same mean and standard deviation of pixel intensities, the same area, and were aligned such that the cross-correlation of each face to a mean face of the database was maximal. Faces were presented at an angle of 15 degrees and were illuminated primarily with ambient light together with an additional but weak point source at 65 degrees azimuth and 25 degree eccentricity.

To show the viability of our approach we require a data set with a good representation of the notion of facial similarity, and to ensure that the data set encompasses both extremes of (dis-)similarity. In the absence of a formal theory of facial similarity we hand-selected a set of faces we thought may show the hypothesised effect: Sixteen of the twenty faces were selected because prior studies had shown them to be consistently and correctly categorised as male or female [18]. Three of the remaining four faces were females that previous subjects found very difficult to categorise and labelled them as female or male almost exactly half of the time. The last face was the mean (androgynous) face across the database. Figure 3 shows the twenty faces thus selected.

Prior to the pairwise comparisons all subjects viewed all twenty faces *simultaneously* arranged in a 4 x 5 grid on the experimental monitor. The subjects were asked to inspect the entire set of faces to obtain a general notion of the relative similarity of the faces and they were instructed to use the entire scale in the following rating task. Subjects were allowed to view the stimuli for however long they wanted. Only thereafter did they proceed to the actual similarity rating stage. Pairwise comparisons of twenty faces requires $\binom{20}{2} = 190$ trials; each of our four subjects completed four repetitions resulting in a total of 760 trials per subject.

During the rating stage faces were shown in pairs in random order for a total duration of 4 seconds (200 msec fade-in, 3600 msec full contrast view, 200 msec fade-out). Subjects were allowed to respond as fast as the wished but had to respond within 5 seconds, i.e. 1 second after the faces had disappeared at the very latest. Similarity was rated on a discrete integer scale between 1 (very dissimilar) and 5 (very similar). The final similarity rating per subject was the mean of the four repetitions within a single subject.

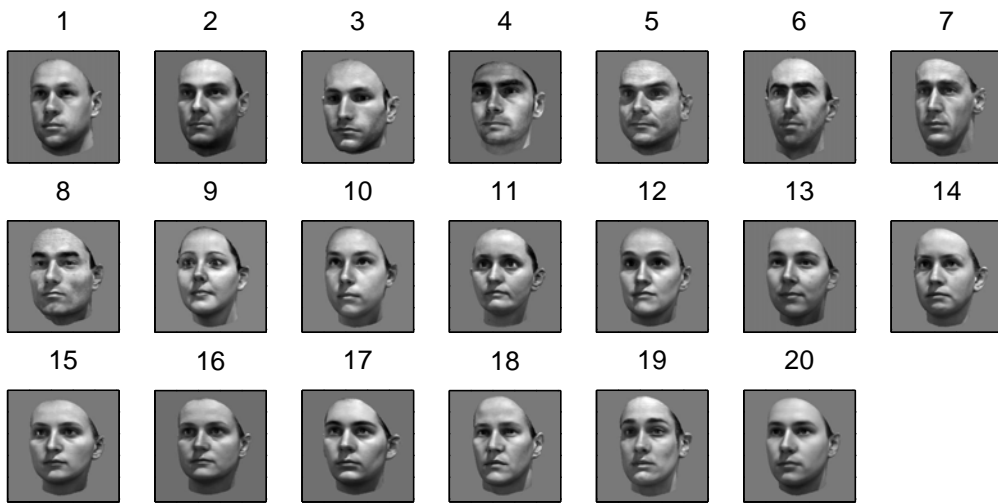

Figure 3: Our data set: Faces 1 to 8 are unambiguous males, faces 9 to 16 are unambiguous females. Faces 17 to 19 are ambiguous and have been attributed to either sex in roughly half the cases. Face 20 is a mean face.

All stimuli were presented on a carefully linearised Siemens SMM21106LS gray-scale monitor with $1024 \times 768$ resolution at a refresh rate of 130Hz driven by a Cambridge Research Systems Visage graphics controller using purpose-written software. The mean luminance of the display was $213\,\mathrm{cd/m^2}$ and presentation of the stimuli did not change the mean luminance of the display.

Three subjects with normal or corrected-to-normal vision—naive to the purpose of the experiment—acted as observers; they were paid for their participation.

We will discuss in detail the results obtained with the first subject. The results from the other subjects are summarized.

In order to exhibit how a single conflictual judgment can break metricity, we follow a two-fold procedure: we first chose a data set of unambiguous faces whose dissimilarities are Euclidean or essential Euclidean. Second, we compare this subsets of faces to a set with those very same unambiguous males and females extended by one additional conflict generating face creating (see Figure 4 for a *schematic* illustration).

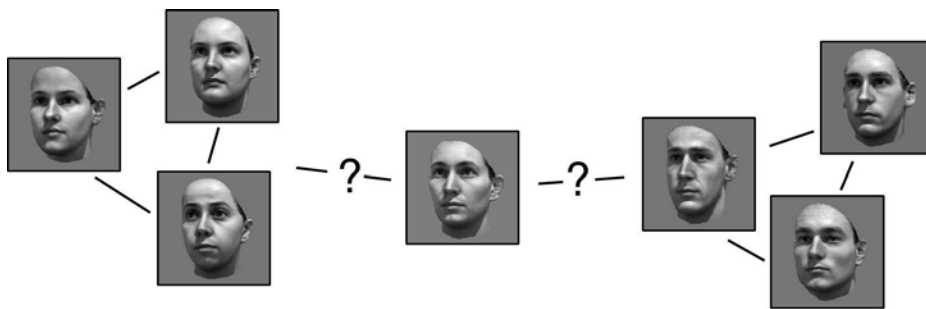

Figure 4: The unambiguous females and unambiguous males lead to a pairwise dissimilarity matrix which is essentially Euclidean. The addition of one single conflicting face introduces the $l_2$ violations.

### 3.1 Subject 1

We chose a subset of faces which has the property that their mutual dissimilarities are essentially Euclidean (Fig. 5). The conflict generating face is 19 and will be denoted as $X$. Fig. 5 shows that the set of unambiguous faces is essentially Euclidean: the smallest eigenvalues of the spectrum are almost zero. This reflects in an almost singular projection in the eigenspace spanned by the eigenvectors associated to the negative eigenvalues. The projection onto the eigenspace spanned by the eigenvectors associated to the positive eigenvalues separates males from females which corresponds to the unique salient feature in the data set.

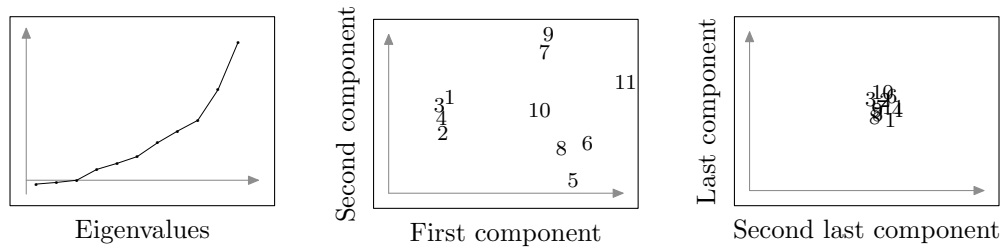

Figure 5: Left: Spectrum with only minor $l_2$ violations, Middle: males vs. females. Right: when a metric is (essentially) Euclidean, the points are concentrated on a singularity in the negative eigenspace.

In order to provoke the conflictual situation, we add one single conflicting face, denoted by $X$. This face has been attributed in previous experiences to either sex in 50 % of the cases. This addition causes the spectrum to flip down, hinting at a unambiguous $l_2$ violation, see Fig. 6. Furthermore, it can be verified that the triangle inequality is violated in several instances by addition of this conflicting judgment reflecting that violation indeed is metric in this case.

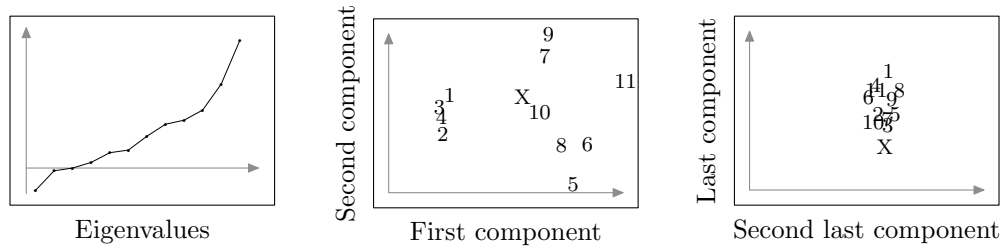

Figure 6: Left: Spectrum with $l_2$-violations, Middle: males vs. females. Right: The conflicting face $X$ is separated from the bulk of faces corresponding to the Euclidean dissimilarities.

The positive projection remains almost unchanged and again separates male from female faces with $X$ in between, reflecting its intermediate position between the males and the females. In the negative projection the $X$ can be seen as separating of the bulk of points which are mutually Euclidean. This corresponds to the effect, albeit not as pronounced, described in the proof of concept illustration 2.1. *Thus we see that the introduction of a conflicting face within a coherent set of unambiguous faces is the cause of the metric violation.*

### 3.2 Subject 2 and 3

The same procedure was applied to the similarity judgments given by Subject 2 and 3. Since the individual perceptual states are incommensurable between different subjects (the reason why we do not average over subjects but only within a subject) the extracted Euclidean subset were different for each of them. However, the process which created the $l_2$-violation *is the same*. Figures 7 and 8 show this process: a conflicting observation destroys the underlying Euclidean structure in the judgements.

Both for Subject 2 and 3 the $X$ lying between the unambiguous faces reflects outside the bulk of Euclidean points concentrated around the singularity in the negative projections.

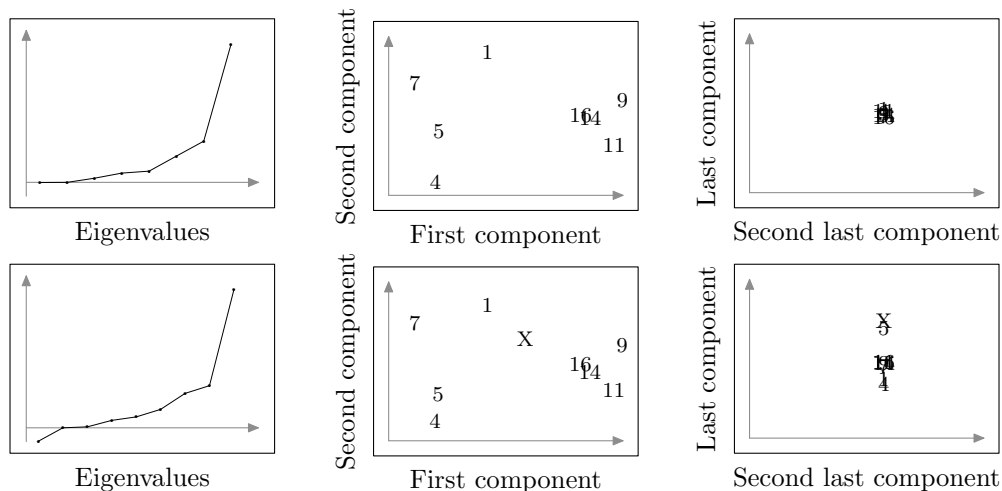

Figure 7: Subject 2: In the upper row, the subset of faces which whose dissimilarities are Euclidean. The lower row shows the effect of introducing a conflicting face $X$ and the subsequent weighting.

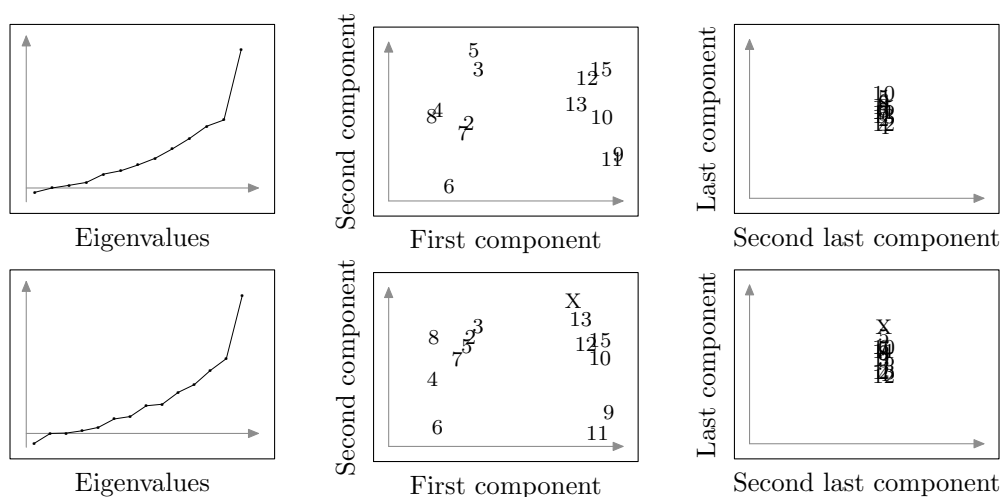

Figure 8: Subject 3: In the upper row, the subset of faces which whose dissimilarities are essentially Euclidean. The lower row shows the effect of introducing a conflicting face $X$ and the subsequent weighting.

Again we obtain that the introduction of a single conflicting face within a set of unambiguous faces for which the human similarity judgment is essentially Euclidean introduces the $l_2$ violations. This strongly corroborates our conflict model and the statement that metric violations in human similarity judgments have a specific meaning, a conflictual judgment for this case.

## 4    Conclusion

We presented a simple experiment in which we could show how a single, purposely selected stimulus introduces $l_2$ violations in what appeared to have been an internal Euclidean similarity space of facial attributes. Importantly, thus, it may not be that there is a clear dichotomy in that internal representations of similarity are either metric or not, rather that they may be for "easy" stimuli but "ambiguous" ones can cause metric violations—at least $l_2$ violations in our setting. We have clearly shown that these violations are caused by conflictual points in a data set: the addition of one such point caused the spectra of the Gram matrices to "flip down" reflecting the $l_2$ violation.

Further research will involve the acquisition of more pairwise similarity judgements in conflicting situations as well as the refinement of our existing experiments. In particular, we would like to know whether it is possible to create larger, scalable conflicts, i.e. conflicts which lead to a much stronger re-weighting and thus to clearer separation of the conflicting point from the bulk of Euclidean points.

## Footnotes

[1]The MPI face database is located at `http://faces.kyb.tuebingen.mpg.de`

## References

[1] F. Gregory Ashby and W. William Lee. Predicting similarity and categorization from identification. *Journal of Experimental Psychology: General*, 120(2):150–172, 1991.

[2] L. Goldfarb. A unified approach to pattern recognition. *Pattern Recognition*, 17:575–582, 1984.

[3] J.K. Kruschke. ALCOVE: an exemplar-based connectionist model of category learning. *Psychological Review*, 99(1):22–44, 1992.

[4] J.B. Kruskal. Multidimensional scaling by optimizing goodness of fit to a nonmetric hypothesis. *Psychometrika*, 29(1):1–27, 1964.

[5] J. Laub and K.-R. Müller. Feature discovery in non-metric pairwise data. *Journal of Machine Learning*, 5:801–818, 2004.

[6] D.L. Medin, R.L. Goldstone, and D. Gentner. Respects for similarity. *Psychological Review*, 100(2):254–278, 1993.

[7] S. Mika, B. Schölkopf, A.J. Smola, K.-R. Müller, M. Scholz, and G. Rätsch. Kernel PCA and de–noising in feature spaces. In M.S. Kearns, S.A. Solla, and D.A. Cohn, editors, *Advances in Neural Information Processing Systems*, volume 11, pages 536–542. MIT Press: Cambridge, MA, 1999.

[8] R.M. Nosofsky. Attention, similarity, and the indentification-categorization relationship. *Journal of Experimental Psychology: General*, 115:39–57, 1986.

[9] E. Pękalska, P. Paclík, and R. P. W. Duin. A generalized kernel approach to dissimilarity-based classification. *Journal of Machine Learning Research*, 2:175–211, 2001.

[10] B. Schölkopf, A. Smola, and K.-R. Müller. Nonlinear component analysis as a kernel eigenvalue problem. *Neural Computation*, 10:1299–1319, 1998.

[11] R. N. Shepard. The analysis of proximities: Multidimensional scaling with an unknown distance function. *Psychometrika*, 27(2):125–140, 1962.

[12] R.N. Shepard. Stimulus and response generalization: A stochastic model relating generalization to distance in psychological space. *Psychometrika*, 22:325–345, 1957.

[13] R.N. Shepard. Toward a universal law of generalization for psychological science. *Science*, 237(4820):1317–1323, 1987.

[14] Roger N. Shepard, Carl I. Hovland, and Herbert M. Jenkins. Learning and memorization of classifications. *Psychological Monographs*, 75(13):1–42, 1961.

[15] W. S. Torgerson. *Theory and Methods of Scaling*. John Wiley and Sons, New York, 1958.

[16] A. Tversky. Features of similarity. *Psychological Review*, 84(4):327–352, 1977.

[17] A. Tversky and I. Gati. Similarity, separability, and the triangle inequality. *Psychological Review*, 89(2):123–154, 1982.

[18] F. A. Wichmann, A. B. A. Graf, E. P. Simoncelli, H. H. Bülthoff, and B. Schölkopf. Machine learning applied to perception: decision-images for classification. In *Advances in Neural Information Processing Systems 17*, pages 1489–1496. MIT Press, 2005.
